# A New View of Automatic Relevance Determination

**David Wipf and Srikantan Nagarajan,** *
Biomagnetic Imaging Lab, UC San Francisco
{david.wipf, sri}@mrsc.ucsf.edu

## Abstract

Automatic relevance determination (ARD) and the closely-related sparse Bayesian learning (SBL) framework are effective tools for pruning large numbers of irrelevant features leading to a sparse explanatory subset. However, popular update rules used for ARD are either difficult to extend to more general problems of interest or are characterized by non-ideal convergence properties. Moreover, it remains unclear exactly how ARD relates to more traditional MAP estimation-based methods for learning sparse representations (e.g., the Lasso). This paper furnishes an alternative means of expressing the ARD cost function using auxiliary functions that naturally addresses both of these issues. First, the proposed reformulation of ARD can naturally be optimized by solving a series of re-weighted $\ell_1$ problems. The result is an efficient, extensible algorithm that can be implemented using standard convex programming toolboxes and is guaranteed to converge to a local minimum (or saddle point). Secondly, the analysis reveals that ARD is exactly equivalent to performing standard MAP estimation in weight space using a particular feature- and noise-dependent, *non-factorial* weight prior. We then demonstrate that this implicit prior maintains several desirable advantages over conventional priors with respect to feature selection. Overall these results suggest alternative cost functions and update procedures for selecting features and promoting sparse solutions in a variety of general situations. In particular, the methodology readily extends to handle problems such as non-negative sparse coding and covariance component estimation.

## 1 Introduction

Here we will be concerned with the generative model

$$\boldsymbol{y} = \Phi\boldsymbol{x} + \boldsymbol{\epsilon}, \tag{1}$$

where $\Phi \in \mathbb{R}^{n \times m}$ is a dictionary of features, $\boldsymbol{x} \in \mathbb{R}^m$ is a vector of unknown weights, $\boldsymbol{y}$ is an observation vector, and $\boldsymbol{\epsilon}$ is uncorrelated noise distributed as $\mathcal{N}(\boldsymbol{\epsilon}; 0, \lambda I)$. When large numbers of features are present relative to the signal dimension, the estimation problem is fundamentally ill-posed. Automatic relevance determination (ARD) addresses this problem by regularizing the solution space using a parameterized, data-dependent prior distribution that effectively prunes away redundant or superfluous features [10]. Here we will describe a special case of ARD called sparse Bayesian learning (SBL) that has been very successful in a variety of applications [15]. Later in Section 4 we will address extensions to more general models.

The basic ARD prior incorporated by SBL is $p(\boldsymbol{x}; \boldsymbol{\gamma}) = \mathcal{N}(\boldsymbol{x}; 0, \text{diag}[\boldsymbol{\gamma}])$, where $\boldsymbol{\gamma} \in \mathbb{R}_+^m$ is a vector of $m$ non-negative hyperparameters governing the prior variance of each unknown coefficient. These hyperparameters are estimated from the data by first marginalizing over the coefficients $\boldsymbol{x}$ and then performing what is commonly referred to as evidence maximization or type-II maximum likelihood [7, 10, 15]. Mathematically, this is equivalent to minimizing

$$\mathcal{L}(\boldsymbol{\gamma}) \triangleq -\log \int p(\boldsymbol{y}|\boldsymbol{x})p(\boldsymbol{x}; \boldsymbol{\gamma})d\boldsymbol{x} = -\log p(\boldsymbol{y}; \boldsymbol{\gamma}) \equiv \log |\Sigma_y| + \boldsymbol{y}^T \Sigma_y^{-1} \boldsymbol{y}, \tag{2}$$

where a flat hyperprior on $\boldsymbol{\gamma}$ is assumed, $\Sigma_y \triangleq \lambda I + \Phi\Gamma\Phi^T$, and $\Gamma \triangleq \mathrm{diag}[\boldsymbol{\gamma}]$. Once some $\boldsymbol{\gamma}_* = \arg\min_{\boldsymbol{\gamma}} \mathcal{L}(\boldsymbol{\gamma})$ is computed, an estimate of the unknown coefficients can be obtained by setting $\boldsymbol{x}_{\mathrm{ARD}}$ to the posterior mean computed using $\boldsymbol{\gamma}_*$:

$$\boldsymbol{x}_{\mathrm{ARD}} = \mathrm{E}[\boldsymbol{x}|\boldsymbol{y}; \boldsymbol{\gamma}_*] = \Gamma_* \Phi^T \Sigma_{y*}^{-1} \boldsymbol{y}. \tag{3}$$

Note that if any $\gamma_{*,i} = 0$, as often occurs during the learning process, then $x_{\mathrm{ARD},i} = 0$ and the corresponding feature is effectively pruned from the model. The resulting weight vector $\boldsymbol{x}_{\mathrm{ARD}}$ is therefore sparse, with nonzero elements corresponding with the 'relevant' features.

There are (at least) two outstanding issues related to this model which we consider to be significant. First, while several methods exist for optimizing (2), limitations remain in each case. For example, an EM version operates by treating the unknown $\boldsymbol{x}$ as hidden data, leading to the E-step

$$\Sigma \triangleq \mathrm{Cov}[\boldsymbol{x}|\boldsymbol{y}; \boldsymbol{\gamma}] = \Gamma - \Gamma\Phi^T\Sigma_y^{-1}\Phi\Gamma, \qquad \boldsymbol{\mu} \triangleq \mathrm{E}[\boldsymbol{x}|\boldsymbol{y}; \boldsymbol{\gamma}] = \Gamma\Phi^T\Sigma_y^{-1}\boldsymbol{y}, \tag{4}$$

and the M-step

$$\gamma_i \to \mu_i^2 + \Sigma_{ii}, \qquad \forall i = 1, \ldots, m. \tag{5}$$

While convenient to implement, the convergence can be prohibitively slow in practice. In contrast, the MacKay update rules are considerably faster to converge [15]. The idea here is to form the gradient of (2), equate to zero, and then form the fixed-point update

$$\gamma_i \to \frac{\mu_i^2}{1 - \gamma_i^{-1}\Sigma_{ii}}, \qquad \forall i = 1, \ldots, m. \tag{6}$$

However, neither the EM nor MacKay updates are guaranteed to converge to a local minimum or even a saddle point of $\mathcal{L}(\boldsymbol{\gamma})$; both have fixed points whenever a $\gamma_i = 0$, whether at a minimizing solution or not. Finally, a third algorithm has recently been proposed that optimally updates a single hyperparameter $\gamma_i$ at a time, which can be done very efficiently in closed form [16]. While extremely fast to implement, as a greedy-like method it can sometimes be more prone to becoming trapped in local minima when the number of features is large, e.g., $m > n$ (results will be presented in a forthcoming publication). Additionally, none of these methods are easily extended to more general problems such as non-negative sparse coding, covariance component estimation, and classification without introducing additional approximations.

A second issue pertaining to the ARD model involves its connection with more traditional *maximum a posteriori* (MAP) estimation methods for extracting sparse, relevant features using fixed, sparsity promoting prior distributions (i.e., heavy-tailed and peaked). Presently, it is unclear how ARD, which invokes a parameterized prior and transfers the estimation problem to hyperparameter space, relates to MAP approaches which operate directly in $\boldsymbol{x}$ space. Nor is it intuitively clear why ARD often works better in selecting optimal feature sets.

This paper introduces an alternative formulation of the ARD cost function using auxiliary functions that naturally addresses the above issues. In Section 2, the proposed reformulation of ARD is conveniently optimized by solving a series of re-weighted $\ell_1$ problems. The result is an efficient algorithm that can be implemented using standard convex programming methods and is guaranteed to converge to a local minimum (or saddle point) of $\mathcal{L}(\boldsymbol{\gamma})$. Section 3 then demonstrates that ARD is exactly equivalent to performing standard MAP estimation in weight space using a particular feature- and noise-dependent, *non-factorial* weight prior. We then show that this implicit prior maintains several desirable advantages over conventional priors with respect to feature selection. Additionally, these results suggest modifications of ARD for selecting relevant features and promoting sparse solutions in a variety of general situations. In particular, the methodology readily extends to handle problems involving non-negative sparse coding, covariance component estimation, and classification as discussed in Section 4.

## 2   ARD/SBL Optimization via Iterative Re-Weighted Minimum $\ell_1$

In this section we re-express $\mathcal{L}(\boldsymbol{\gamma})$ using auxiliary functions which leads to an alternative update procedure that circumvents the limitations of current approaches. In fact, a wide variety of alternative update rules can be derived by decoupling $\mathcal{L}(\boldsymbol{\gamma})$ using upper bounding functions that are more conveniently optimized. Here we focus on a particular instantiation of this idea that leads to an iterative minimum $\ell_1$ procedure. The utility of this selection being that many powerful convex programming toolboxes have already been developed for solving these types of problems, especially when structured dictionaries $\Phi$ are being used.

## 2.1 Algorithm Derivation

To start we note that the log-determinant term of $\mathcal{L}(\boldsymbol{\gamma})$ is concave in $\boldsymbol{\gamma}$ (see Section 3.1.5 of [1]), and so can be expressed as a minimum over upper-bounding hyperplanes via

$$\log|\Sigma_y| = \min_{\boldsymbol{z}} \boldsymbol{z}^T\boldsymbol{\gamma} - g^*(\boldsymbol{z}), \tag{7}$$

where $g^*(\boldsymbol{z})$ is the concave conjugate of $\log|\Sigma_y|$ that is defined by the duality relationship [1]

$$g^*(\boldsymbol{z}) = \min_{\boldsymbol{\gamma}} \boldsymbol{z}^T\boldsymbol{\gamma} - \log|\Sigma_y|, \tag{8}$$

although for our purposes we will never actually compute $g^*(\boldsymbol{z})$. This leads to the following upper-bounding auxiliary cost function

$$\mathcal{L}(\boldsymbol{\gamma},\boldsymbol{z}) \triangleq \boldsymbol{z}^T\boldsymbol{\gamma} - g^*(\boldsymbol{z}) + \boldsymbol{y}^T\Sigma_y^{-1}\boldsymbol{y} \geq \mathcal{L}(\boldsymbol{\gamma}). \tag{9}$$

For any fixed $\boldsymbol{\gamma}$, the optimal (tightest) bound can be obtained by minimizing over $\boldsymbol{z}$. The optimal value of $\boldsymbol{z}$ equals the slope at the current $\boldsymbol{\gamma}$ of $\log|\Sigma_y|$. Therefore, we have

$$\boldsymbol{z}_{\text{opt}} = \nabla_{\boldsymbol{\gamma}}\log|\Sigma_y| = \text{diag}\left[\Phi^T\Sigma_y^{-1}\Phi\right]. \tag{10}$$

This formulation naturally admits the following optimization scheme:

Step 1: Initialize each $z_i$, e.g., $z_i = 1, \forall i$.

Step 2: Solve the minimization problem

$$\boldsymbol{\gamma} \to \arg\min_{\boldsymbol{\gamma}} \mathcal{L}_{\boldsymbol{z}}(\boldsymbol{\gamma}) \triangleq \boldsymbol{z}^T\boldsymbol{\gamma} + \boldsymbol{y}^T\Sigma_y^{-1}\boldsymbol{y}. \tag{11}$$

Step 3: Compute the optimal $\boldsymbol{z}$ using (10).

Step 4: Iterate Steps 2 and 3 until convergence to some $\boldsymbol{\gamma}_*$.

Step 5: Compute $\boldsymbol{x}_{\text{ARD}} = \text{E}[\boldsymbol{x}|\boldsymbol{y};\boldsymbol{\gamma}_*] = \Gamma_*\Phi^T\Sigma_{y*}^{-1}\boldsymbol{y}$.

**Lemma 1.** The objective function in (11) is convex.

This can be shown using Example 3.4 and Section 3.2.2 in [1]. Lemma 1 implies that many standard optimization procedures can be used for the minimization required by Step 2. For example, one attractive option is to convert the problem to an equivalent *least absolute shrinkage and selector operator* or 'Lasso' [14] optimization problem according to the following:

**Lemma 2.** The objective function in (11) can be minimized by solving the weighted convex $\ell_1$-regularized cost function

$$\boldsymbol{x}_* = \arg\min_{\boldsymbol{x}} \|\boldsymbol{y} - \Phi\boldsymbol{x}\|_2^2 + 2\lambda\sum_i z_i^{1/2}|x_i| \tag{12}$$

and then setting $\gamma_i \to z_i^{-1/2}|x_{*,i}|$ for all $i$ (note that each $z_i$ will always be positive).

The proof of Lemma 2 can be briefly summarized using a re-expression of the data dependent term in (11) using

$$\boldsymbol{y}^T\Sigma_y^{-1}\boldsymbol{y} = \min_{\boldsymbol{x}} \frac{1}{\lambda}\|\boldsymbol{y} - \Phi\boldsymbol{x}\|_2^2 + \sum_i \frac{x_i^2}{\gamma_i}. \tag{13}$$

This leads to an upper-bounding auxiliary function for $\mathcal{L}_{\boldsymbol{z}}(\boldsymbol{\gamma})$ given by

$$\mathcal{L}_{\boldsymbol{z}}(\boldsymbol{\gamma},\boldsymbol{x}) \triangleq \sum_i \left(z_i\gamma_i + \frac{x_i^2}{\gamma_i}\right) + \frac{1}{\lambda}\|\boldsymbol{y} - \Phi\boldsymbol{x}\|_2^2 \geq \mathcal{L}_{\boldsymbol{z}}(\boldsymbol{\gamma}), \tag{14}$$

which is jointly convex in $\boldsymbol{x}$ and $\boldsymbol{\gamma}$ (see Example 3.4 in [1]) and can be globally minimized by solving over $\boldsymbol{\gamma}$ and then $\boldsymbol{x}$. For any $\boldsymbol{x}$, $\gamma_i = z_i^{-1/2}|x_i|$ minimizes $\mathcal{L}_{\boldsymbol{z}}(\boldsymbol{\gamma},\boldsymbol{x})$. When substituted into (14) we obtain (12). When solved for $\boldsymbol{x}$, the global minimum of (14) yields the global minimum of (11) via the stated transformation.

In summary then, by iterating the above algorithm using Lemma 2 to implement Step 2, a convenient optimization method is obtained. Moreover, we do not even need to globally solve for $\boldsymbol{x}$ (or equivalently $\boldsymbol{\gamma}$) at each iteration as long as we strictly reduce (11) at each iteration. This is readily achievable using a variety of simple strategies. Additionally, if $\boldsymbol{z}$ is initialized to a vector of ones, then the starting point (assuming Step 2 is computed in full) is the exact Lasso estimator. The algorithm then refines this estimate through the specified re-weighting procedure.

## 2.2 Global Convergence Analysis

Let $\mathcal{A}(\cdot)$ denote a mapping that assigns to every point in $\mathbb{R}_+^m$ the subset of $\mathbb{R}_+^m$ which satisfies Steps 2 and 3 of the proposed algorithm. Such a mapping can be implemented via the methodology described above. We allow $\mathcal{A}(\cdot)$ to be a point-to-set mapping to handle the case where the global minimum of (11) is not unique, which could occur, for example, if two columns of $\Phi$ are identical.

**Theorem 1.** From any initialization point $\boldsymbol{\gamma}_{(0)} \in \mathbb{R}_+^m$ the sequence of hyperparameter estimates $\{\boldsymbol{\gamma}_{(k)}\}$ generated via $\boldsymbol{\gamma}_{(k+1)} \in \mathcal{A}(\boldsymbol{\gamma}_{(k+1)})$ is guaranteed to converge monotonically to a local minimum (or saddle point) of (2).

The proof is relatively straightforward and stems directly from the *Global Convergence Theorem* (see for example [6]). A sketch is as follows: First, it must be shown that the the mapping $\mathcal{A}(\cdot)$ is compact. This condition is satisfied because if any element of $\boldsymbol{\gamma}$ is unbounded, $\mathcal{L}(\boldsymbol{\gamma})$ diverges to infinity. If fact, for any fixed $\boldsymbol{y}$, $\Phi$ and $\lambda$, there will always exist a radius $r$ such that for any $\|\boldsymbol{\gamma}_{(0)}\| \leq r$, $\|\boldsymbol{\gamma}_{(k)}\| \leq r$ for all $k$. Second, we must show that for any non-minimizing point of $\mathcal{L}(\boldsymbol{\gamma})$ denoted $\boldsymbol{\gamma}'$, $\mathcal{L}(\boldsymbol{\gamma}'') < \mathcal{L}(\boldsymbol{\gamma}')$ for all $\boldsymbol{\gamma}'' \in \mathcal{A}(\boldsymbol{\gamma}')$. At any non-minimizing $\boldsymbol{\gamma}'$ the auxiliary cost function $\mathcal{L}_{\boldsymbol{z}'}(\boldsymbol{\gamma})$ obtained from Step 3 will be strictly tangent to $\mathcal{L}(\boldsymbol{\gamma})$ at $\boldsymbol{\gamma}'$. It will therefore necessarily have a minimum elsewhere since the slope at $\boldsymbol{\gamma}'$ is nonzero by definition. Moreover, because the $\log|\cdot|$ function is strictly concave, at this minimum the actual cost function will be reduced still further. Consequently, the proposed updates represent a valid descent function. Finally, it must be shown that $\mathcal{A}(\cdot)$ is closed at all non-stationary points. This follows from related arguments. The algorithm could of course theoretically converge to a saddle point, but this is rare and any minimal perturbation leads to escape.

Both EM and MacKay updates provably fail to satisfy one or more of the above criteria and so global convergence cannot be guaranteed. With EM, the failure occurs because the associated updates do not always strictly reduce $\mathcal{L}(\boldsymbol{\gamma})$. Rather, they only ensure that $\mathcal{L}(\boldsymbol{\gamma}'') \leq \mathcal{L}(\boldsymbol{\gamma}')$ at all points. In contrast, the MacKay updates do not even guarantee cost function decrease. Consequently, both methods can become trapped at a solution such as $\boldsymbol{\gamma} = 0$; a fixed point of the updates but not a stationary point or local minimum of $\mathcal{L}(\boldsymbol{\gamma})$. However, in practice this seems to be more of an issue with the MacKay updates. Related shortcomings of EM in this regard can be found in [19]. Finally, the fast Tipping updates could potentially satisfy the conditions for global convergence, although this matter is not discussed in [16].

## 3 Relating ARD to MAP Estimation

In hierarchical models such as ARD and SBL there has been considerable debate over how to best perform estimation and inference [8]. Do we add a hyperprior and then integrate out $\boldsymbol{\gamma}$ and perform MAP estimation directly on $\boldsymbol{x}$? Or is it better to marginalize over the coefficients $\boldsymbol{x}$ and optimize the hyperparameters $\boldsymbol{\gamma}$ as we have described in this paper? In specific cases, arguments have been made for the merits of one over the other based on intuition or heuristic arguments [8, 15]. But we would argue that this distinction is somewhat tenuous because, as we will now show using ideas from the previous section, the weights obtained from the ARD type-II ML procedure can equivalently be viewed as arising from an explicit MAP estimate in $\boldsymbol{x}$ space. This notion is made precise as follows:

**Theorem 2.** Let $\boldsymbol{x}^2 \triangleq [x_1^2, \ldots, x_m^2]^T$ and $\boldsymbol{\gamma}^{-1} \triangleq [\gamma_1^{-1}, \ldots, \gamma_m^{-1}]^T$. Then the ARD coefficients from (3) solve the MAP problem

$$\boldsymbol{x}_{\text{ARD}} = \arg\min_{\boldsymbol{x}} \|\boldsymbol{y} - \Phi\boldsymbol{x}\|_2^2 + \lambda h^*(\boldsymbol{x}^2), \tag{15}$$

where $h^*(\boldsymbol{x}^2)$ is the concave conjugate of $h(\boldsymbol{\gamma}^{-1}) \triangleq -\log|\Sigma_y|$ and is a concave, non-decreasing function of $\boldsymbol{x}$.

This result can be established using much of the same analysis used in previous sections. Omitting some details for the sake of brevity, using (13) we can create a strict upper bounding auxiliary function on $\mathcal{L}(\boldsymbol{\gamma})$:

$$\mathcal{L}(\boldsymbol{\gamma}, \boldsymbol{x}) = \frac{1}{\lambda}\|\boldsymbol{y} - \Phi\boldsymbol{x}\|_2^2 + \sum_i \frac{x_i^2}{\gamma_i} + \log|\Sigma_y|. \tag{16}$$

If we optimize first over $\boldsymbol{\gamma}$ instead of $\boldsymbol{x}$ (allowable), the last two terms form the stated concave conjugate function $h^*(\boldsymbol{x}^2)$. In turn, the minimizing $\boldsymbol{x}$, which solves (15), is identical to that obtained by ARD. The concavity of $h^*(\boldsymbol{x}^2)$ with respect each $|x_i|$ follows from similar ideas.

**Corollary 1.** The regularization term in (15), and hence the implicit prior distribution on $\boldsymbol{x}$ given by $p(\boldsymbol{x}) \propto \exp[-\frac{1}{2}h^*(\boldsymbol{x}^2)]$, is not generally factorable, meaning $p(\boldsymbol{x}) \neq \prod_i p_i(x_i)$. Additionally, unlike traditional MAP procedures (e.g., Lasso, ridge regression, etc.), this prior is explicitly dependent on both the dictionary $\Phi$ and the regularization term $\lambda$.

This result stems directly from the fact that $h(\boldsymbol{\gamma}^{-1})$ is non-factorable and is dependent on $\Phi$ and $\lambda$. The only exception occurs when $\Phi^T\Phi = I$; here $h^*(\boldsymbol{x}^2)$ factors and can be expressed in closed form independently of $\Phi$, although $\lambda$ dependency remains.

### 3.1 Properties of the implicit ARD prior

To begin at the most superficial level, the $\Phi$ dependency of the ARD prior leads to scale invariant solutions, meaning the value of $\boldsymbol{x}_{\mathrm{ARD}}$ is not affected if we rescale $\Phi$, i.e., $\Phi \rightarrow \Phi D$, where $D$ is a diagonal matrix. Rather, any rescaling $D$ only affects the implicit initialization of the algorithm, not the shape of the cost function.

More significantly, the ARD prior is particularly well-designed for finding sparse solutions. We should note that concave, non-decreasing regularization functions are well-known to encourage sparse representations. Since $h^*(\boldsymbol{x}^2)$ is such a function, it should therefore not be surprising that it promotes sparsity to some degree. However, when selecting highly sparse subsets of features, the factorial $\ell_0$ quasi-norm is often invoked as the ideal regularization term given unlimited computational resources. It is expressed via $\|\boldsymbol{x}\|_0 \triangleq \sum_i I[x_i \neq 0]$, where $I[\cdot]$ denotes the indicator function, and so represents a count of the number of nonzero coefficients (and therefore features). By applying a $\exp[-1/2(\cdot)]$ transformation, we obtain the implicit (improper) prior distribution. The associated MAP estimation problem (assuming the same standard Gaussian likelihood) involves solving

$$\min_{\boldsymbol{x}} \|\boldsymbol{y} - \Phi\boldsymbol{x}\|_2^2 + \lambda\|\boldsymbol{x}\|_0. \tag{17}$$

The difficulty here is that (17) is nearly impossible to solve in general; it is NP-hard owing to a combinatorial number of local minima and so the traditional idea is to replace $\|\cdot\|_0$ with a tractable approximation. For this purpose, the $\ell_1$ norm is the optimal or tightest *convex* relaxation of the $\ell_0$ quasi-norm, and therefore it is commonly used leading to the Lasso algorithm [14]. However, the $\ell_1$ norm need not be the best relaxation in general. In Sections 3.2 and 3.3 we demonstrate that the non-factorable, $\lambda$-dependent $h^*(\boldsymbol{x}^2)$ provides a tighter, albeit *non-convex*, approximation that promotes greater sparsity than $\|\boldsymbol{x}\|_1$ while conveniently producing many fewer local minima than using $\|\boldsymbol{x}\|_0$ directly. We also show that, in certain settings, no $\lambda$-independent, factorial regularization term can achieve similar results. Consequently, the widely used family of $\ell_p$ quasi-norms, i.e., $\|\boldsymbol{x}\|_p \triangleq \sum_i |x_i|^p$, $p < 1$ [2], or the Gaussian entropy measure $\sum_i \log |x_i|$ based on the Jeffreys prior [4] provably fail in this regard.

### 3.2 Benefits of $\lambda$ dependency

To explore the properties of $h^*(\boldsymbol{x}^2)$ regarding $\lambda$ dependency alone, we adopt the simplifying assumption $\Phi^T\Phi = I$. (Later we investigate the benefits of a non-factorial prior.) In this special case, $h^*(\boldsymbol{x}^2)$ is factorable and can be expressed in closed form via

$$h^*(\boldsymbol{x}^2) = \sum_i h^*(x_i^2) \propto \sum_i \frac{2|x_i|}{|x_i| + \sqrt{x_i^2 + 4\lambda}} + \log\left(2\lambda + x_i^2 + |x_i|\sqrt{x_i^2 + 4\lambda}\right), \tag{18}$$

which is independent of $\Phi$. A plot of $h^*(x_i^2)$ is shown in Figure 1 (*left*) below.

The $\lambda$ dependency is retained however and contributes two very desirable properties: (i) As a strictly concave function of each $|x_i|$, $h^*(\boldsymbol{x}^2)$ more closely approximates the $\ell_0$ quasi-norm than the $\ell_1$ norm while, (ii) The associated cost function (15) is unimodal unlike when $\lambda$-independent approximations, e.g., the $\ell_p$ quasi-norm, are used. This can be explained as follows. When $\lambda$ is small, the Gaussian likelihood is highly restrictive, constraining most of its relative mass to a very localized region of $\boldsymbol{x}$ space. Therefore, a tighter prior more closely resembling the $\ell_0$ quasi-norm can be used without the risk of local minima, which occur when the spines of a sparse prior overlap non-negligible portions of the likelihood (see Figure 6 in [15] for a good 2D visual of a sparse prior with characteristic spines running alone the coordinate axis). In the limit as $\lambda \rightarrow 0$, $h^*(\boldsymbol{x}^2)$ converges to a scaled version of the

$\ell_0$ quasi-norm, yet no local minimum exist because the likelihood in this case only permits a single feasible solution with $\boldsymbol{x} = \Phi^T \boldsymbol{y}$. In contrast, when $\lambda$ is large, the likelihood is less constrained and a looser prior is required to avoid local minima troubles, which will arise whenever the now relatively diffuse likelihood intersects the sharp spines of a highly sparse prior. In this situation $h^*(\boldsymbol{x}^2)$ more closely resembles a scaled version of the $\ell_1$ norm. The implicit ARD prior naturally handles this transition becoming sparser as $\lambda$ decreases and vice versa. Hence the following property, which is easy to show [18]:

**Lemma 3.** When $\Phi^T\Phi = I$, (15) has no local minima whereas (17) has $2^M$ local minima.

Use of the $\ell_1$ norm in place of $h^*(\boldsymbol{x}^2)$ also yields no local minima; however, it is a much looser approximation of $\ell_0$ and penalizes coefficients linearly unlike $h^*(\boldsymbol{x}^2)$. The benefits of $\lambda$ dependency in this regard can be formalized and will be presented in a subsequent paper. As a final point of comparison, the actual weight estimate obtained from solving (15) when $\Phi^T\Phi = I$ is equivalent to the non-negative garrote estimator that has been advocated for wavelet shrinkage [5, 18].

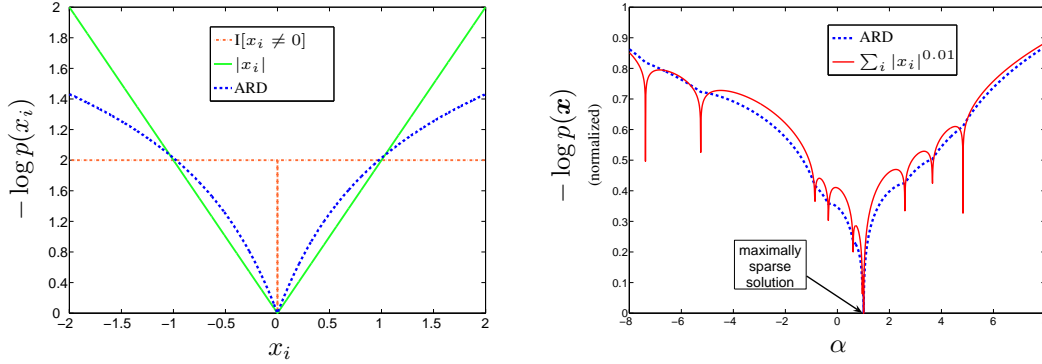

Figure 1: *Left*: 1D example of the implicit ARD prior. The $\ell_1$ and $\ell_0$ norms are included for comparison. *Right*: Plot of the ARD prior across the feasible region as parameterized by $\alpha$. A factorial prior given by $-\log p(\boldsymbol{x}) \propto \sum_i |x_i|^{0.01} \approx \|\boldsymbol{x}\|_0$ is included for comparison. Both approximations to the $\ell_0$ norm retain the correct global minimum, but only ARD smooths out local minima.

### 3.3  Benefits of a non-factorial prior

In contrast, the benefits the typically non-factorial nature of $h^*(\boldsymbol{x}^2)$ are most pronounced when $m > n$, meaning there are more features than the signal dimension $\boldsymbol{y}$. In a noiseless setting (with $\lambda \to 0$), we can explicitly quantify the potential of this property of the implicit ARD prior. In this limiting situation, the canonical sparse MAP estimation problem (17) reduces to finding

$$\boldsymbol{x}_0 \triangleq \arg\min_{\boldsymbol{x}} \|\boldsymbol{x}\|_0 \quad \text{s.t. } \boldsymbol{y} = \Phi\boldsymbol{x}. \tag{19}$$

By simple extension of results in [18], the global minimum of (15) in the limit as $\lambda \to 0$ will equal $\boldsymbol{x}_0$, assuming the latter is unique. The real distinction then is regarding the number of local minimum. In this capacity the ARD MAP problem is superior to any possible factorial variant:

**Theorem 3.** In the limit as $\lambda \to 0$ and assuming $m > n$, no *factorial* prior $p(\boldsymbol{x}) = \prod_i \exp[-1/2 f_i(x_i)]$ exists such that the corresponding MAP problem $\min_{\boldsymbol{x}} \|\boldsymbol{y} - \Phi\boldsymbol{x}\|_2^2 + \lambda \sum_i f_i(x_i)$ is: (i) Always globally minimized by a maximally sparse solution $\boldsymbol{x}_0$ and, (ii) Has fewer local minima than when solving (15).

A sketch of the proof is as follows. First, for any factorial prior and associated regularization term $\sum_i f_i(x_i)$, the only way to satisfy (i) is if $\partial f_i(x_i)/\partial x_i \to \infty$ as $x_i \to 0$. Otherwise, it will always be possible to have a $\Phi$ and $\boldsymbol{y}$ such that $\boldsymbol{x}_0$ is not the global minimum. It is then straightforward to show that any $f_i(x_i)$ with this property will necessarily have between $\left[\binom{m-1}{n} + 1, \binom{m}{n}\right]$ local minimum. Using results from [18], this is provably an upper bound on the number of local minimum to (15). Moreover, with the exception of very contrived situations, the number of ARD local minima will be considerably less. In general, this result speaks directly to the potential limitations of restricting oneself to factorial priors when maximal feature pruning is paramount.

While generally difficult to visualize, in restricted situations it is possible to explicitly illustrate the type of smoothing over local minima that is possible using non-factorial priors. For example,

consider the case where $m = n + 1$ and $\text{Rank}(\Phi) = n$, implying that $\Phi$ has a null-space dimension of one. Consequently, any feasible solution to $\boldsymbol{y} = \Phi\boldsymbol{x}$ can be expressed as $\boldsymbol{x} = \boldsymbol{x}' + \alpha\boldsymbol{v}$, where $\boldsymbol{v} \in \text{Null}(\Phi)$, $\alpha$ is any real-valued scalar, and $\boldsymbol{x}'$ is any fixed, feasible solution (e.g., the minimum norm solution). We can now plot any prior distribution $p(\boldsymbol{x})$, or equivalently $-\log p(\boldsymbol{x})$, over the 1D feasible region of $\boldsymbol{x}$ space as a function of $\alpha$ to view the local minima profile.

To demonstrate this idea, we chose $n = 10$, $m = 11$ and generated a $\Phi$ matrix using iid $\mathcal{N}(0, 1)$ entries. We then computed $\boldsymbol{y} = \Phi\boldsymbol{x}_0$, where $\|\boldsymbol{x}_0\|_0 = 9$ and nonzero entries are also iid unit Gaussian. Figure 1 (*right*) displays the plots of two example priors in the feasible region of $\boldsymbol{y} = \Phi\boldsymbol{x}$: (i) the non-factorial implicit ARD prior, and (ii) the prior $p(\boldsymbol{x}) \propto \exp(-\frac{1}{2}\sum_i |x_i|^p)$, $p = 0.01$. The later is a factorial prior which converges to the ideal sparsity penalty when $p \to 0$. From the figure, we observe that, while both priors peak at the $\boldsymbol{x}_0$, the ARD prior has substantially smoothed away local minima. While the implicit Lasso prior (which is equivalent to the assumption $p = 1$) also smooths out local minima, the global minimum may be biased away from the maximally sparse solution in many situations, unlike the ARD prior which provides a non-convex approximation with its global minimum anchored at $\boldsymbol{x}_0$.

## 4 Extensions

Thus far we have restricted attention to one particularly useful ARD-based model. But much of the analysis can be extended to handle a variety of alternative data likelihoods and priors. A particularly useful adaptation relevant to compressed sensing [17], manifold learning [13], and neuroimaging [12, 18] is as follows. First, the data $\boldsymbol{y}$ can be replaced with a $n \times t$ observation matrix $Y$ which is generated via an unknown coefficient matrix $X$. The assumed likelihood model and prior are

$$p(Y|X) \propto \exp\left(-\frac{1}{2\lambda}\|Y - \Phi X\|_{\mathcal{F}}^2\right), \quad p(X) \propto \exp\left(-\frac{1}{2}\text{trace}\left[X^T \Sigma_x^{-1} X\right]\right), \quad \Sigma_x \triangleq \sum_{i=1}^{d_\gamma} \gamma_i C_i.$$
(20)

Here each of the $d_\gamma$ matrices $C_i$'s are known covariance components of which the irrelevant ones are pruned by minimizing the analogous type-II likelihood function

$$\mathcal{L}(\boldsymbol{\gamma}) = \log|\lambda I + \Phi\Sigma_x\Phi^T| + \text{trace}\left[\frac{1}{t}XX^T\left(\lambda I + \Phi\Sigma_x\Phi^T\right)^{-1}\right].$$
(21)

With minimal effort, this extension can be solved using the methodology described herein. The primary difference is that Step 2 becomes a second-order cone (SOC) optimization problem for which a variety of techniques exist for its minimization [2, 9].

Another very useful adaptation involves adding a non-negativity constraint on the coefficients $\boldsymbol{x}$, e.g., non-negative sparse coding. This is easily incorporated into the MAP cost function (15) and optimization problem (12); performance is often significantly better than the non-negative Lasso. Results will be presented in a subsequent paper. It may also be possible to develop an effective variant for handling classification problems that avoids additional approximations such as those introduced in [15].

## 5 Discussion

While ARD-based approaches have enjoyed remarkable success in a number of disparate fields, they remain hampered to some degree by implementational limitations and a lack of clarity regarding the nature of the cost function and existing update rules. This paper addresses these issues by presenting a principled alternative algorithm based on auxiliary functions and a dual representation of the ARD objective. The resulting algorithm is initialized at the well-known Lasso solution and then iterates via a globally convergent re-weighted $\ell_1$ procedure that in many ways approximates ideal subset selection using the $\ell_0$ norm. Preliminary results using this methodology on toy problems as well as large neuroimaging simulations with $m \approx 100,000$ are very promising (and will be reported in future papers). A good (highly sparse) solution is produced at every iteration and so early stopping is always feasible if desired. This produces a highly efficient, global competition among features that is potentially superior to the sequential (greedy) updates of [16] in terms of local minima avoidance in certain cases when $\Phi$ is highly overcomplete (i.e., $m \gg n$). Moreover, it is also easily extended to handle additional constraints (e.g., non-negativity) or model complexity as occurs with general covariance component estimation. A related optimization strategy has also been reported in [3].

The analysis used in deriving this algorithm reveals that ARD is exactly equivalent to performing MAP estimation in $x$ space using a principled, sparsity-inducing prior that is non-factorable and dependent on both the feature set and noise parameter. We have shown that these qualities allow it to promote maximally sparse solutions at the global minimum while relenting drastically fewer local minima than competing priors. This might possibly explain the superior performance of ARD/SBL over Lasso in a variety of disparate disciplines where sparsity is crucial [11, 12, 18]. These ideas raise a key question: If we do not limit ourselves to factorable, $\Phi$- and $\lambda$-independent regularization terms/priors as is commonly done, then what is the optimal prior $p(x)$ in the context of feature selection? Perhaps there is a better choice that does not neatly fit into current frameworks linked to empirical priors based on the Gaussian distribution. Note that the $\ell_1$ re-weighting scheme for optimization can be applied to a broad family of non-factorial, sparsity-inducing priors.

## Footnotes

*This research was supported by NIH grants R01DC04855 and R01DC006435.

## References

[1] S. Boyd and L. Vandenberghe, *Convex Optimization*, Cambridge University Press, 2004.

[2] S.F. Cotter, B.D. Rao, K. Engan, and K. Kreutz-Delgado, "Sparse solutions to linear inverse problems with multiple measurement vectors," *IEEE Trans. Signal Processing*, vol. 53, no. 7, pp. 2477–2488, April 2005.

[3] M. Fazel, H. Hindi, and S. Boyd "Log-Det Heuristic for Matrix Rank Minimization with Applications to Hankel and Euclidean Distance Matrices," *Proc. American Control Conf.*, vol. 3, pp. 2156–2162, June 2003.

[4] M.A.T. Figueiredo, "Adaptive sparseness using Jeffreys prior," *Advances in Neural Information Processing Systems 14*, pp. 697–704, 2002.

[5] H. Gao, "Wavelet shrinkage denoising using the nonnegative garrote," *Journal of Computational and Graphical Statistics*, vol. 7, no. 4, pp. 469–488, 1998.

[6] D.G. Luenberger, *Linear and Nonlinear Programming*, Addison–Wesley, Reading, Massachusetts, 2nd ed., 1984.

[7] D.J.C. MacKay, "Bayesian interpolation," *Neural Comp.*, vol. 4, no. 3, pp. 415–447, 1992.

[8] D.J.C. MacKay, "Comparison of approximate methods for handling hyperparameters," *Neural Comp.*, vol. 11, no. 5, pp. 1035–1068, 1999.

[9] D.M. Malioutov, M. Çetin, and A.S. Willsky, "Sparse signal reconstruction perspective for source localization with sensor arrays," *IEEE Trans. Signal Processing*, vol. 53, no. 8, pp. 3010–3022, August 2005.

[10] R.M. Neal, *Bayesian Learning for Neural Networks*, Springer-Verlag, New York, 1996.

[11] R. Pique-Regi, E.S. Tsau, A. Ortega, R.C. Seeger, and S. Asgharzadeh, "Wavelet footprints and sparse Bayesian learning for DNA copy number change analysis," *Int. Conf. Acoustics Speech and Signal Processing*, April 2007.

[12] R.R. Ramírez, *Neuromagnetic Source Imaging of Spontaneous and Evoked Human Brain Dynamics*, PhD Thesis, New York University, 2005.

[13] J.G. Silva, J.S. Marques, and J.M. Lemos, "Selecting landmark points for sparse manifold learning," *Advances in Neural Information Processing Systems 18*, pp. 1241–1248, 2006.

[14] R. Tibshirani, "Regression shrinkage and selection via the Lasso," *Journal of the Royal Statistical Society*, vol. 58, no. 1, pp. 267–288, 1996.

[15] M.E. Tipping, "Sparse Bayesian learning and the relevance vector machine," *Journal of Machine Learning Research*, vol. 1, pp. 211–244, 2001.

[16] M.E. Tipping and A.C. Faul, "Fast marginal likelihood maximisation for sparse Bayesian models," *Ninth Int. Workshop Artificial Intelligence and Statistics*, Jan. 2003.

[17] M.B. Wakin, M.F. Duarte, S. Sarvotham, D. Baron, and R.G. Baraniuk, "Recovery of jointly sparse signals from a few random projections," *Advances in Neural Information Processing Systems 18*, pp. 1433–1440, 2006.

[18] D.P. Wipf, "Bayesian Methods for Finding Sparse Representations," PhD Thesis, UC San Diego, 2006.

[19] C.F. Wu, "On the convergence properties of the EM algorithm," *The Annals of Statistics*, vol. 11, pp. 95–103, 1983.
